# Mixtures of Controllers for Jump Linear and Non-linear Plants

**Timothy W. Cacciatore**
Department of Neurosciences
University of California at San Diego
La Jolla, CA 92093

**Steven J. Nowlan**
Synaptics, Inc.
2698 Orchard Parkway
San Jose, CA 95134

## Abstract

We describe an extension to the Mixture of Experts architecture for modelling and controlling dynamical systems which exhibit multiple modes of behavior. This extension is based on a Markov process model, and suggests a recurrent network for gating a set of linear or non-linear controllers. The new architecture is demonstrated to be capable of learning effective control strategies for jump linear and non-linear plants with multiple modes of behavior.

## 1  Introduction

Many stationary dynamic systems exhibit significantly different behaviors under different operating conditions. To control such complex systems it is computationally more efficient to decompose the problem into smaller subtasks, with different control strategies for different operating points. When detailed information about the plant is available, gain scheduling has proven a successful method for designing a global control (Shamma and Athans, 1992). The system is partitioned by choosing several operating points and a linear model for each operating point. A controller is designed for each linear model and a method for interpolating or 'scheduling' the gains of the controllers is chosen.

The control problem becomes even more challenging when the system to be controlled is non-stationary, and the mode of the system is not explicitly observable. One important, and well studied, class of non-stationary systems are jump linear systems of the form: $\frac{dx}{dt} = A(i)x + B(i)u.$ where $x$ represents the system state,

$u$ the input, and $i$, the stochastic parameter that determines the mode of the system, is not explicitly observable. To control such a system, one must estimate the mode of the system from the input-output behavior of the plant and then choose an appropriate control strategy.

For many complex plants, an appropriate decomposition is not known *a priori*. One approach is to learn the decomposition and the piecewise solutions in parallel. The Mixture of Experts architecture (Nowlan 1990, Jacobs *et al* 1991) was proposed as one approach to simultaneously learning a task decomposition and the piecewise solutions in a neural network context. This architecture has been applied to control simple stationary plants, when the operating mode of the plant was explicitly available as an input to the gating network (Jacobs and Jordan 1991).

There is a problem with extending this architecture to deal with non-stationary systems such as jump linear systems. The original formulation of this architecture was based on an assumption of statistical independence of training pairs appropriate for classification tasks. However, this assumption is inappropriate for modelling the causal dependencies in control tasks. We derive an extension to the original Mixture of Experts architecture which we call the Mixture of Controllers. This extension is based on an $n$th order Markov model and can be implemented to control non-stationary plants. The new derivation suggests the importance of using recurrence in the gating network, which then learns to estimate the conditional state occupancy for sequences of outputs. The power of the architecture is illustrated by learning control and switching strategies for simple jump linear and non-stationary non-linear plants. The modified recurrent architecture is capable of learning both the control and switching for these plants. while a non-recurrent architecture fails to learn an adequate control.

## 2   Mixtures of Controllers

The architecture of the system is shown in figure 1. $x_t$ denotes the vector of inputs to the controller at time $t$ and $y_t$ is the corresponding overall control output. The architecture is identical to the Mixture of Experts architecture, except that the gating network has become recurrent, receiving its outputs from the previous time step as part of its input. The underlying statistical model, and corresponding training procedure for the Mixture of Controllers, is quite different from that originally proposed for the Mixture of Experts.

We assume that the system we are interested in controlling has $N$ different modes or states[1] and we will have a distinct control $M_k$ for each mode. In general we are interested in the likelihood of producing a sequence of control outputs $y_1, \ldots, y_T$ given a sequence of inputs $x_1, \ldots, x_T$. This likelihood can be computed as:

$$
\begin{aligned}
L(y_1 \ldots y_T | x_1 \ldots x_T) &= \prod_t \sum_k P(y_t | s_t = k, x_t) P(s_t = k | y_1 \ldots y_{t-1}, x_1 \ldots x_t) \\
&= \prod_t \sum_k b_t^k \gamma_t^k \quad (1)
\end{aligned}
$$

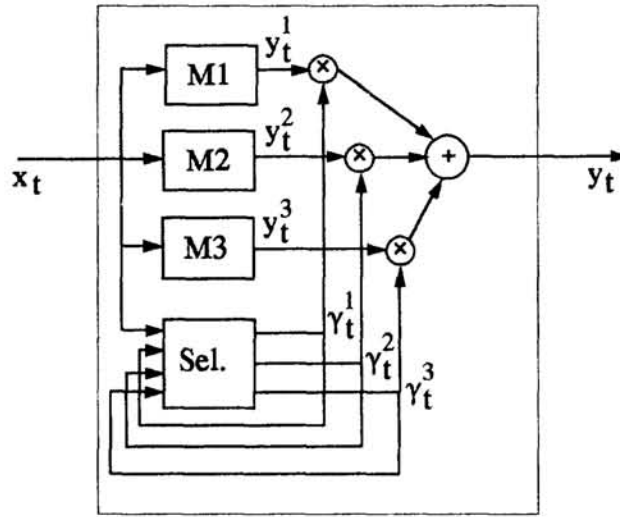

Figure 1: The Mixture of Controllers architecture. **M1**, **M2** and **M3** are feedforward networks implementing controls appropriate for different modes of the system to be controlled. The gating network (**Sel.**) is recurrent and uses a softmax non-linearity to compute the weight to be assigned to each of the control outputs. The weighted sum of the controls is then used as the overall control for the plant.

where $b_t^k$ represents the probability of producing the desired control $y_t$ given the input $x_t$ and that the system is in state $k$. $\gamma_t^k$ represents the conditional probability of being in state $k$ given the sequence of inputs and outputs seen so far. In order to make the problem tractable, we assume that this conditional probability is completely determined by the current input to the system and the previous state of the system:

$$\gamma_t^k = f_{w_\gamma}(x_t, \{\gamma_{t-1}^j\}).$$

Thus we are assuming that our control can be approximated by a Markov process, and since we are assuming that the mode of the system is not explicitly available, this becomes a hidden Markov model. This Markov assumption leads to the particular recurrent gating architecture used in the Mixture of Controllers.

If we make the same gaussian assumptions used in the original Mixture of Experts model, we can define a gradient descent procedure for maximizing the log of the likelihood given in Equation 1. Assume

$$b_t^k = \frac{1}{\sqrt{2\pi}\sigma} e^{-(y_t - y_t^k)^2/2\sigma^2}$$

and define $\beta_t^k = P(y_T, \ldots, y_t | s_k, x_T, \ldots, x_t)$, $L_t = \sum_k \beta_t^k \gamma_t^k$ and

$$R_t^k = \frac{\beta_t^k \gamma_t^k}{L_t}.$$

Then the derivative of the likelihood with respect to the output of one of the controllers becomes:

$$\frac{\partial \log L}{\partial y_t^k} = K R_t^k (y_t - y_t^k). \tag{2}$$

The derivative of the likelihood with respect to a weight in one of the control networks is computed by accumulating partial derivatives over the sequence of control outputs:

$$\frac{\partial \log L}{\partial w_q} = \sum_t \frac{\partial \log L}{\partial y_t^k} \frac{\partial y_t^k}{\partial w_q}. \tag{3}$$

For the gating network, we once again use a softmax non-linearity so:

$$\gamma_t^k = \frac{\exp g_t^k}{\sum_j \exp g_t^j}.$$

Then

$$\frac{\partial \log L}{\partial g_t^k} = \sum_t (R_t^k - \gamma_t^k)\gamma_{t-1}^k. \tag{4}$$

The derivatives for the weights in the gating network are again computed by accumulating partial derivatives over output sequences:

$$\frac{\partial \log L}{\partial w_r} = \sum_t \frac{\partial \log L}{\partial g_t^k} \frac{\partial g_t^k}{\partial w_r}. \tag{5}$$

Equations (2) and (4) turn out to be quite similar to those derived for the original Mixture of Experts architecture. The primary difference is the appearance of $\beta_t^k$ rather than $b_t^k$ in the expression for $R_t^k$. The appearance of $\beta$ is a direct result of the recurrence introduced into the gating network. $\beta$ can be computed as part of a modified back propagation through time algorithm for the gating network using the recurrence:

$$\beta_t^k = b_t^k + \sum_j \omega_{kj}\beta_{t+1}^j \tag{6}$$

where

$$\omega_{kj} = \frac{\partial \gamma_{t+1}^j}{\partial \gamma_t^k}$$

Equation (6) is the analog of the backward pass in the forward-backward algorithm for standard hidden Markov models.

In the simulations reported in the next section, we used an online gradient descent procedure which employs an approximation for $\beta_t^k$ which uses only one step of back propagation through time. This approximation did not appear to significantly affect the final performance of the recurrent architecture.

## 3    Results

The performances of the recurrent Mixture of Controllers and non-recurrent Mixture of Experts were compared on three control tasks: a first order jump linear system, a second order jump linear system, and a tracking task that required two non-linear controllers. The object of the first two jump-linear tasks was to control a plant which switched randomly between two linear systems. The resulting overall systems were highly non-linear. In both the first and second order cases it was

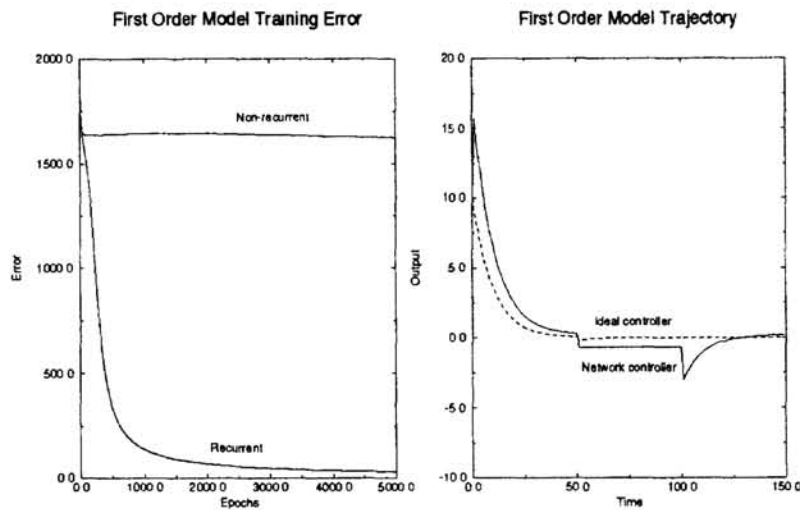

Figure 2: **Left**: Training convergence of Mixtures of Experts and Mixtures of Controllers on first order jump linear system. The vertical axis is average squared error over training sequences and horizontal axis is the number of training sequences seen. **Right**: Sample test trajectory of first order jump linear system under control of Mixture of Controllers. The system switches states at times 50 and 100.

desired to drive all plant outputs to zero (zero-forcing control). Neither the first or second order systems could be successfully controlled by a single linear controller.

For both jump-linear tasks, the architecture of the Mixture of Controllers and Mixture of Experts consisted of two linear experts, and a one layer gating network. The input to the experts was the plant output at the previous time step, while the input to the gating network was the ratio of the plant outputs at the two preceding time steps. An ideal linear controller was designed for each mode of the system. Training targets were derived from outputs of the appropriate ideal controller, using the known mode of the system for the training trajectories. The parameters of the gating and control networks were updated after each pass through sample trajectories which contained several state transitions.

The recurrent Mixture of Controllers could be trained to successfully control the first order jump linear system (figure 2), and once trained generalized successfully to novel test trajectories. The non-recurrent Mixture of Experts failed to learn even the training data for the first order jump linear system (note the high asymptote for the training error without recurrence in figure 2). The recurrent Mixture of Controllers was also able to learn to control the second order jump linear system (figure 3), however, it was necessary to teacher force the system during the first 5000 epochs of training by providing the true mode of the system as an extra input to the gating network. This extra input was removed at epoch 5000 and the error initially increases dramatically but the system is able to eventually learn to control the second order jump linear system autonomously. Note that the Mixture of Experts system is actually able to learn a successful control even more rapidly than the Mixture of Controllers when the additional teacher input is provided, however learning again completely fails once this input is removed at epoch 5000 (figure 3).

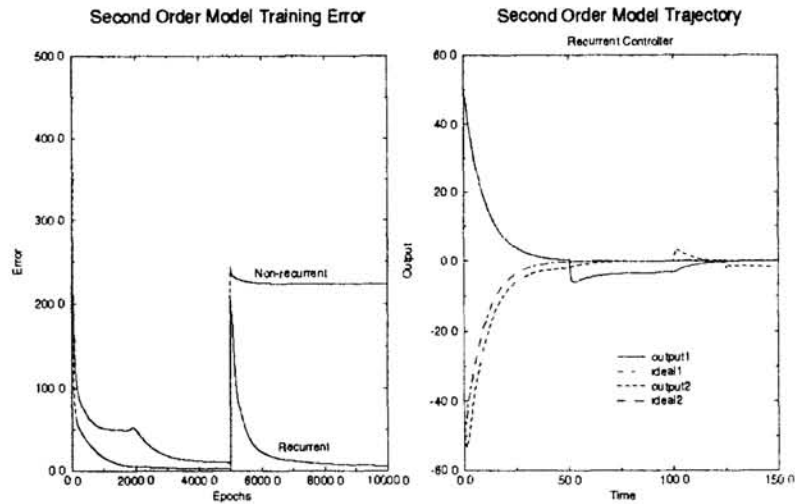

Figure 3: **Left**: Training convergence of Mixtures of Experts and Mixtures of Controllers on second order jump linear system. **Right**: Sample test trajectory of second order jump linear system under control of Mixture of Controllers. The system again switches states at times 50 and 100.

In both first and second order cases, the trained Mixture of Controllers is able to control the system in both modes of system behavior, and to detect mode changes automatically. The difficulty in designing a control for a jump linear system usually lies in identifying the state of the system. No explicit law describing how to identify and switch between control modes is necessary to train the Mixture of Controllers, as this is learned automatically as a byproduct of learning to successfully control the system.

Performance of the Mixture of Controllers and the Mixture of Experts was also compared on a more complex task requiring a non-linear control law in each mode. The task was to control the trajectory of a ship to track an object traveling in a straight line, or flee from an object having a random walk trajectory (figure 4). There is a high degree of task interference between the controls appropriate during the two modes of object behaviors. The ship dynamics were taken from Miller and Sutton (1990).

For both the Mixture of Controllers and the Mixture of Experts two experts were used. The experts received past and present measurements of the object bearing, distance, velocity, and the ship heading and turn rate. The controllers specified the desired turn rate of the ship. A one layer gating network was used which received the velocity of the object as input.

Training targets were produced from ideal controllers designed for each object behavior. The ideal controller for the random walk behavior produced a turn rate that headed directly away from the object. The ideal controller for intercepting the object used future information about object position to determine the turn rate which would lead to the closest possible intercept point. Both ideal controllers made use of information not available to the Mixture of Experts or Mixture of Controllers.

The Mixture of Controllers and the Mixture of Experts were trained on sequences of

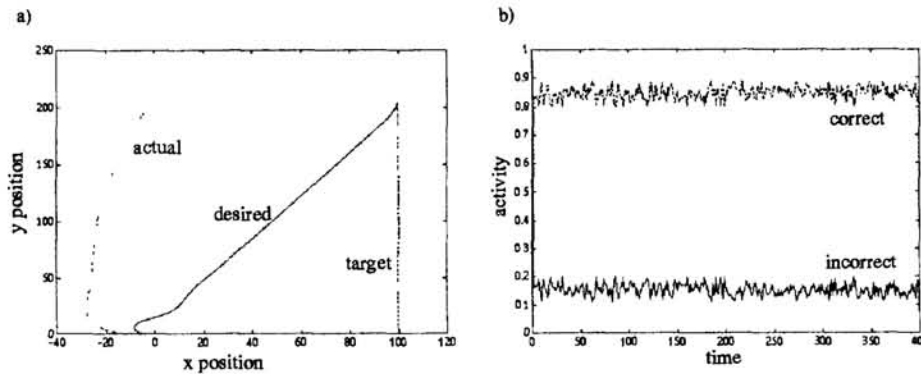

Figure 4: (a) Actual and desired trajectories of ship under control of Mixture of Experts while attempting to intercept target. (b) Gating unit activities as a function of time for trajectory in (a).

trajectories where the object changed behaviors multiple times. The weights of the networks were updated after each pass through the trajectories. The input to the gating net in this task provided more instantaneous information about the mode of object behavior than was provided in the jump linear tasks. As a result, the non-recurrent Mixture of Experts was able to achieve a minimum level of performance on the overall task. The recurrent Mixture of Controllers performed much better.

The differences between two architectures are revealed by examining the gating network outputs. Without recurrence, the Mixture of Experts gating network could not determine the state of the object with certainty, and compromised by selecting a combination of the correct and incorrect control (figure 4b). Since the two controls are incompatible, this uncertainty degrades the performance of the overall controller. With recurrence in the gating network, the Mixture of Controllers is able to determine the target state with greater certainty by integrating information from many observations of object behavior. The sharper decisions about which control to use greatly improve tracking performance (figure 5).

We explored the ability of the Mixture of Controllers to learn the dynamics of switching by training on trajectories where the object switched behavior with varying frequency. The gating network trained on an object that switched behaviors infrequently was sluggish to respond to transitions, but more noise tolerant than the gating network trained on a frequently switching object. Thus, the gating network is able to incorporate the frequency of transition into its state model.

## 4   Discussion

We have described an extension to the Mixture of Experts architecture for modelling and controlling dynamical systems which exhibit multiple modes of behavior. The algorithm we have presented for updating the parameters of the model is a simple gradient descent procedure. Application of the technique to large scale problems

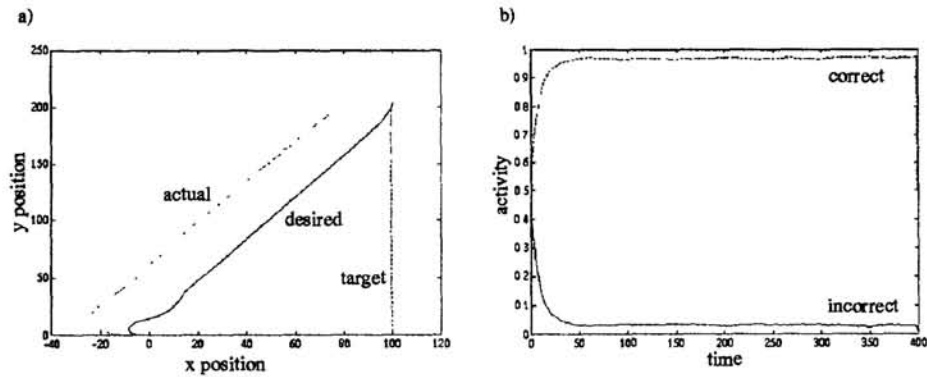

Figure 5: (a) Actual and desired trajectories of ship under control of Mixture of Controllers while attempting to intercept target. (b) Gating unit activities as a function of time for trajectory in (a). Note that these are much less noisy than the activities seen in figure 4(b).

may require the development of faster converging update algorithms, perhaps based on the generalized EM (GEM) family of algorithms, or a variant of the iterative reweighted least squares procedure proposed by Jordan and Jacobs (1993) for hierarchies of expert networks. Additional work is also required to establish the stability and convergence rate of the algorithm for use in adaptive control applications.

## Footnotes

[1]This is an idealization and if $N$ is unknown it is safest to overestimate it.

## References

Jacobs, R.A. and Jordan, M.I. A competitive modular connectionist architecture. *Neural Information Processing Systems* 3 (1991).

Jacobs, R.A., Jordan, M.I., Nowlan, S.J. and Hinton, G.E. Adaptive Mixtures of Local Experts. *Neural Computation*, **3**, 79-87, (1991).

Jordan, M.I. and Jacobs, R.A. Hierarchical Mixtures of Experts and the EM algorithm. *Neural Computation*, (1994).

Miller, W.T., Sutton, R.S. and Werbos, P.J. *Neural Networks for Control*, MIT Press (1993).

Nowlan, S.J. Competing Experts: An Experimental Investigation of Associative Mixture Models. *Technical Report CRG-TR-90-5*, Department of Computer Science, University of Toronto (1990).

Shamma, J.S., and Athans, M. Gain scheduling: potential hazards and possible remedies. *IEEE Control Systems Magazine*, 12:(3), 101-107 (1992).
